# Probabilistic Belief Revision with Structural Constraints

**Peter B. Jones**
MIT Lincoln Laboratory
Lexington, MA 02420
jonep@ll.mit.edu

**Venkatesh Saligrama**
Dept. of ECE
Boston University
Boston, MA 02215
srv@bu.edu

**Sanjoy K. Mitter**
Dept. of EECS
MIT
Cambridge, MA 02139
mitter@mit.edu

## Abstract

Experts (human or computer) are often required to assess the probability of uncertain events. When a collection of experts independently assess events that are structurally interrelated, the resulting assessment may violate fundamental laws of probability. Such an assessment is termed incoherent. In this work we investigate how the problem of incoherence may be affected by allowing experts to specify likelihood models and then update their assessments based on the realization of a globally-observable random sequence.

**Keywords:** Bayesian Methods, Information Theory, consistency

## 1 Introduction

Coherence is perhaps the most fundamental property of probability estimation. Coherence will be formally defined later, but in essence a coherent probability assessment is one that exhibits logical consistency. Incoherent assessments are those that cannot be correct, that are at odds with the underlying structure of the space, and so can't be extended to a complete probability distribution [1, 2]. From a decision theoretic standpoint, treating assessments as odds, incoherent assessments result in guaranteed losses to assessors. They are dominated strategies, meaning that for every incoherent assessment there is a coherent assessment that uniformly improves the outcome for the assessors. Despite this fact, expert assessments (human and machine) are vulnerable to incoherence [3].

Previous authors have used coherence as a tool for fusing distributed expert assessments [4, 5, 6]. The focus has been on *static* coherence in which experts are polled once about some set of events and the responses are then fused through a geometric projection. Besides relying on arbitrary scoring functions to define the "right" projection, such analyses don't address dynamically evolving assessments or forecasts. This paper is, to our knowledge, the first attempt to analyze the problem of coherence under Bayesian belief dynamics. The importance of dynamic coherence is demonstrated in the following example.

Consider two uncertain events $A_1$ and $A_2$ where $A_1 \subseteq A_2$ (e.g. $A_2 = \{\text{NASDAQ} \uparrow \text{tomorrow}\}$ and $A_1 = \{\text{NASDAQ} \uparrow \text{tomorrow} \geq 10 \text{ points}\}$). To be coherent, a probability assessment must obey the relation $P(A_1) \leq P(A_2)$. For the purposes of the example, suppose the initial belief is $P(A_1) = P(A_2) = 0.5$ which is coherent. Next, suppose there is some binary random variable

---

This work was sponsored by the U.S. Government under Air Force Contract FA8721-05-C-0002. Opinions, interpretations, conclusions, and recommendations are those of the authors and are not necessarily endorsed by the United States Government

$Z$ that is believed to correlate with the underlying event (e.g. $Z = \mathbb{1}_{\{\text{Google}\uparrow\text{today}\}}$ where $\mathbb{1}$ is an indicator function). The believed dependence between $Z$ and $A_i$ is captured by a likelihood model $P(Z|A_i)$ that gives the probability of observing $Z$ when event $A_i$ does or doesn't occur. For the example, suppose $Z = 0$ and the believed likelihoods are $P(Z = 0|A_1) = 1$ and $P(Z = 0|\bar{A}_1)$ and $P(Z = 0|A_2) = P(Z = 0|\bar{A}_2) = 0.5$ where $\bar{A}$ is the complement of $A$. There's nothing inherently irrational in this belief model, but when Bayes' Rule is applied, it gives $P(A_1|Z = 0) = 0.67 > P(A_2|Z = 0) = 0.5$. The belief update has *introduced* incoherence!

## 1.1 Motivating Example

Concerned with their network security, BigCorps wants to purchase an Intrusion Detection and Prevention System (IDPS). They have two options, IDPS$_1$ and IDPS$_2$. IDPS$_1$ detects both distributed denial of service (DDoS) attacks and port scan (PS) attacks, while IDPS$_2$ detects only DDoS attacks. While studying the NIST guide to IDPSs [7], BigCorps' CTO notes the recommendation that "organizations should consider using multiple types of IDPS technologies to achieve more comprehensive and accurate detection and prevention of malicious activity." Following the NIST recommendation, BigCorps purchases both IDPSs and sets them to work monitoring network traffic.

One morning while reading the output reports of the two detectors, an intrepid security analyst witnesses an interesting behavior. IDPS$_2$ is registering an attack probability of $0.1$ while detector IDPS$_1$ is reading an attack probability of $0.05$. Since the threats detected by IDPS$_1$ are a superset of those detected by IDPS$_2$, the probability assigned by IDPS$_1$ should always be larger than that assigned by IDPS$_2$. The dilemma faced by our analyst is how to reconcile the logically incoherent outputs of the two detectors. Particularly, how to ascribe probabilities in a way that is logically consistent, but still retains as much as possible the expert assessments of the detectors.

## 1.2 Contributions of this Work

This work introduces the concept of dynamic coherence, one that has not been previously treated in the literature. We suggest two possible forms of dynamic coherence and analyze the relationship between them. They are implemented and compared in a simple network modeling simulation.

## 1.3 Previous Work

Previous authors have analyzed coherence with respect to contingent (or conditional) probability assessments [8, 9, 10]. These developments attempt to determine conditions characterizing coherent subjective posteriors. While likelihood models are a form of contingent probability assessment, this paper goes further in analyzing the impact of these assessments on coherent belief dynamics.

In [11, 12] a different form of conditional coherence is suggested which derives from coherence of a joint probability distribution over observations and states of nature. It is shown that for this stronger form of conditional coherence, certain specially structured event sets and likelihood functions will produce coherent posterior assessments.

Logical consistency under non-Bayesian belief dynamics has been previously analyzed. In [13] conditions for invariance under permutations of the observational sequence under Jeffrey's rule are developed. A comparison of Jeffrey's rule and Pearl's virtual evidence method is made in [14] which shows that the virtual evidence method implicitly assumes the conditions of Jeffrey's update rule.

## 2 Model

Let $\Omega = \{\omega_1, \omega_2, \ldots\}$ be an event space and $(\Omega, \mathcal{F})$ a measurable space. Let $\theta : \Omega \to \Theta$ be a measurable random variable; consider $\Theta = \{\theta^1, \theta^j, \ldots, \theta^J\}$ to be the set of all possible "states of the world." Also, let $Z_i : \Omega \to \mathcal{Z}$ be a sequence of measureable random variables; consider $Z_i$ to be the sequence of observations, with $\mathcal{Z} = \{z^1, z^2, \ldots, z^K\}$ and $K < \infty$. Let $\Omega_\theta$ (resp. $\Omega_{Z_i}$) be

the pre-image of $\theta$ (resp. $Z_i$). Since the random variables are assumed measureable, $\Omega_\theta$ and $\Omega_{Z_i}$ are measureable sets (i.e. elements of $\mathcal{F}$), as are their countable intersections and unions.

For $i = 1, 2, \ldots, N$, let $A_i^\theta$ be a subset of $\Theta$, let $A_i = \cup_{\{\theta \in A_i^\theta\}} \Omega_\theta$ and let $\mathcal{A} = \{A_i\}$. We call elements of $\mathcal{A}$ **events under assessment**. The characteristic matrix $\chi$ for the events under assessment is defined as

$$\chi_{ij} = \left\{ \begin{array}{ll} 1 & \theta^j \in A_i^\theta \\ 0 & \text{o.w.} \end{array} \right. .$$

An individual probability assessment $P : \mathcal{A} \to [0, 1]$ maps each event under assessment to the unit interval. In an abuse of notation, we will let $P \triangleq \left[ \begin{array}{cccc} P(A_1) & P(A_2) & \cdots & P(A_N) \end{array} \right]^T$ be a (joint) probability assessment. A **coherent** assessment (i.e. one that is logically consistent) can be described geometrically as lying in the convex hull of the columns of $\chi$, meaning $\exists \lambda \in [0, 1]^J$ s.t. $\sum_i \lambda_i = 1$ and $P = \chi \lambda$.

We now consider a sequence of probability assessments $P_n$ defined as follows: $P_n$ is the result of a belief revision process based on an initial probability assessment $P_0$, a likelihood model $p_n(z|A)$, and a sequence of observations $Z_1, Z_2, \ldots, Z_n$. A likelihood model $p_n(z|\mathcal{A})$ is a pair of probability mass functions over the observations: one conditioned on $A$ and the other conditioned on $\bar{A}$ (where $\bar{A}$ denotes the complement of $A$). We will make the simplifying assumption that the likelihood model is *static*, i.e. $p_n(z|A) = p(z|A)$ and $p_n(z|\bar{A}) = p(z|\bar{A})$ for all $n$.

In this paper we assume belief revision dynamics governed by Bayes' rule, i.e.

$$P_{n+1} = \frac{p(z_{n+1}|A) * P_n}{p(z_{n+1}|A) * P_n + p(z_{n+1}|\bar{A}) * (1 - P_n)} = \frac{1}{1 + \frac{p(z_{n+1}|\bar{A})}{p(z_{n+1}|A)} \frac{1 - P_n}{P_n}}$$

To simplify development, denote $p(z = z^i|A_j) = \alpha_{ij}$ and $p(z = z^i|\bar{A}_j) = \beta_{ij}$ and assume $\forall j$, $\exists i$ s.t. $\alpha_{ij} \neq \beta_{ij}$ (i.e. each event has at least one informative observation) and $\alpha_{ij} \in (0, 1)$, $\beta_{ij} \in (0, 1)$ for all $i, j$ (i.e. no observation determines absolutely whether any event obtains). Then by induction the posterior probability of event $A$ after $n$ observations is:

$$P_n(A_j) = \frac{1}{1 + \frac{1 - P_0}{P_0} \prod_{i=1}^{K} \left( \frac{\beta_{ij}}{\alpha_i} \right)^{n_i}} \tag{1}$$

when $n_i$ is the number of observations $z^i$.

## 3  Probability convergence for single assessors

For a single assessor revising his estimate of the likelihood of event $A$, let the probability model be given by $p(z = z^i|A) = \alpha_i$ and $p(z = z^i|\bar{A}) = \beta_i$. It is convenient to rewrite (1) in terms of the ratio $\rho_i = \frac{n_i}{n}$ and for simplicity assuming $P_0 = 0.5$ (although the analysis holds for general $P_0 \in (0, 1)$). Substituting yields

$$P_n = \frac{1}{1 + \left[ \prod_{i=1}^{K} \left( \frac{\beta_i}{\alpha_i} \right)^{\rho_i} \right]^n} \tag{2}$$

Note that 1) $\rho$ is the empirical distribution over the observations, and so converges almost surely (a.s.) to the true generating distribution, and 2) the convergence properties of $P_n$ are determined by the quantity between the square brackets in (2). Specifically, let

$$L_\infty = \lim_{n \to \infty} \prod_{i=1}^{K} \left( \frac{\beta_i}{\alpha_i} \right)^{\rho_i}$$

$L_\infty$ is commonly referred to as the likelihood ratio, familiar from classical binary hypothesis testing. Since $\rho$ converges a.s. and the function is continuous, $L_\infty$ exists a.s. If $L_\infty < 1$ then $P_n \to 1$; if $L_\infty > 1$ then $P_n \to 0$; if $L_\infty = 1$ then $P_n \to \frac{1}{2}$.

## 3.1 Matched likelihood functions

Assume that the likelihood model is both infinitely precise and infinitely accurate, meaning that when $A$ (resp. $\bar{A}$) obtains observations are generated i.i.d. according to $\alpha$ (resp. $\beta$).

Assume that $A$ obtains; then $L_\infty = \prod_{i=1}^{K} \left( \frac{\beta_i}{\alpha_i} \right)^{\alpha_i}$ a.s. Let $\mathcal{L}_\infty = \log L_\infty$ which in this case yields

$$\mathcal{L}_\infty = \log \prod_{i=1}^{K} \left( \frac{\beta_i}{\alpha_i} \right)^{\alpha_i} = \sum_{i=1}^{K} \alpha_i \log \frac{\beta_i}{\alpha_i} = -D(\alpha||\beta) < 0$$

where all relations hold a.s., $D(\cdot||\cdot)$ is the relative entropy [15], and the last inequality follows since by assumption $\alpha \neq \beta$. Since $\mathcal{L}_\infty < 0 \Leftrightarrow L_\infty < 1$, this implies that when the true generating distribution is $\alpha$, $P_n \to 1$ a.s.

Similarly, when $\bar{A}$ obtains, we have

$$\mathcal{L}_\infty = \log \prod_{i=1}^{K} \left( \frac{\beta_i}{\alpha_i} \right)^{\beta_i} = \sum_{i=1}^{K} \beta_i \log \frac{\beta_i}{\alpha_i} = D(\beta||\alpha) > 0$$

and $P_n \to 0$ a.s.

## 3.2 Mismatched likelihood functions

Now consider the situation when the expert assessed likelihood model is incorrect. Assume the observation generating distribution is $\gamma = \mathbb{P}(Z_i = z)$ where $\gamma \neq \alpha$ and $\gamma \neq \beta$. In this case, $\mathcal{L}_\infty = \sum \gamma_i \log \frac{\beta_i}{\alpha_i}$. We define

$$T(\gamma) = -\mathcal{L}_\infty = \sum_i \gamma_i \log \frac{\alpha_i}{\beta_i} \tag{3}$$

Then the probability simplex over the observation space $Z$ can be partitioned into two sets: $\mathcal{P}_0 = \{\gamma | T(\gamma) < 0\}$ and $\mathcal{P}_1 = \{\gamma | T(\gamma) > 0\}$. By the a.s. convergence of the empirical distribtuion, $\gamma \in \mathcal{P}_i \Rightarrow P_n \to i$. (The boundary set $\{\gamma | T(\gamma) = 0\}$ represents an unstable equilibrium in which $P_n$ a.s. converges to $\frac{1}{2}$).

The problem of mismatched likelihood functions is similar to composite hypothesis testing (c.f. [16] and references therein). Composite hypothesis testing attempts to design tests to determine the truth or falsity of a hypothesis with some ambiguity in the underlying parameter space. Because of this ambiguity, each hypothesis $\mathcal{H}_i$ corresponds not to a single distribution, but to a set of possible distributions. In the mismatched likelihood function problem, composite spaces are formed due to the properties of Bayes' rule for a specific likelihood model. A corollary of the above result is that if $\mathcal{H}_i \subseteq \mathcal{P}_i$ then Bayes' rule (under the specific likelihood model) is an asymptotically perfect detector.

## 4 Multiple Assessors with Structural Constraints

In Section 3 we analyzed convergence properties of a single event under assessment. Considering multiple events introduces the challenge of defining a dynamic concept of coherence for the assessment revision process. In this section we suggest two possible definitions of dynamic coherence and consider some of the implications of these definitions.

### 4.1 Step-wise Coherence

We first introduce a step-wise definition of coherence, and derive equivalency conditions for the special class of 2-expert likelihood models.

**Definition 1** *Under the Bayes' rule revision process, a likelihood model $p(z|\mathcal{A})$ is **step-wise coherent (SWC)** if $P_n \in \mathtt{convhull}(\chi) \Rightarrow P_{n+1} \in \mathtt{convhull}(\chi)$ for all $z \in \mathcal{Z}$.*

Essentially this definition says that if the posterior assessment process is coherent at any time, it will remain coherent perpetually, independent of observation sequence. We derive necessary and sufficient conditions for SWC for the characteristic matrix given by

$$\chi = \left[ \begin{array}{ccc} 1 & 1 & 0 \\ 0 & 1 & 0 \end{array} \right] \tag{4}$$

Generalizations of this development are possible for any $\chi \in \{0,1\}^{2 \times |\Theta|}$.

Note that under the characteristic matrix given by (4) a model is SWC iff $P_n(A_1) \geq P_n(A_2)$ for all $n$ and all coherent $P_0$. Proceeding inductively, assume $P_n$ is *marginally* SWC, i.e. $P_n(A_1) = P_n(A_2) = \pi$. Due to the continuity of the update rule, a model will be SWC iff it is coherent at the margins. For coherence, for any $i$ we must have $P_{n+1}(A_1) \geq P_{n+1}(A_2)$. By substitution into (1)

$\frac{\alpha_{i1}\pi}{\alpha_{i1}\pi + \beta_{i1}(1-\pi)} \geq \frac{\alpha_{i2}\pi}{\alpha_{i2}\pi + \beta_{i2}(1-\pi)}$ or, equivalently, $\frac{\alpha_{i1}}{\alpha_{i2}} \geq \frac{\alpha_{i1}\pi + \beta_{i1}(1-\pi)}{\alpha_{i2}\pi + \beta_{i2}(1-\pi)}$.

By monotonicity, $\frac{\alpha_{i1}\pi + \beta_{i1}(1-\pi)}{\alpha_{i2}\pi + \beta_{i2}(1-\pi)} \in \left[ \min\left\{ \frac{\alpha_{i1}}{\alpha_{i2}}, \frac{\beta_{i1}}{\beta_{i2}} \right\}, \max\left\{ \frac{\alpha_{i1}}{\alpha_{i2}}, \frac{\beta_{i1}}{\beta_{i2}} \right\} \right]$. Since $\frac{\alpha_{i1}}{\alpha_{i2}} \geq \frac{\alpha_{i1}}{\alpha_{i2}}$ degenerately, for $\chi$ given by (4), the model will be SWC iff $\frac{\alpha_{i1}}{\alpha_{i2}} \geq \frac{\beta_{i1}}{\beta_{i2}} \, \forall i$, or (rearranging)

$$\forall i, \; \frac{\alpha_{i1}}{\beta_{i1}} \geq \frac{\alpha_{i2}}{\beta_{i2}} \tag{5}$$

## 4.2 Asymptotic coherence

While it is relatively simple to characterize coherent models in the two assessor case, in general SWC is difficult to check. As such, we introduce a simpler condition:

**Definition 2** *A likelihood model $p(z|A)$ is **weakly asymptotically coherent (WAC)** if for all observation generating distributions $\gamma$ s.t. $\lim_{n \to \infty} P_n \in \{0,1\}^N$, $\exists i$ s.t. $\lim_{n \to \infty} P_n = \chi e_i$ a.s., where $e_i$ is the $i^{th}$ unit vector.*

**Lemma 1** *Step-wise coherence implies weakly asymptotic coherence.*

Assume that a model is SWC but not WAC. Since it's not WAC, there exists a $\gamma$ s.t. $Z_i$ drawn IID from $\gamma$ a.s. results in $P_n \to \hat{P}$ where $\hat{P} \in \{0,1\}^N$ is not a column of $\chi$ and is therefore not coherent. Since this holds regardless of initial conditions, assume the process is initialized coherently. Then, by a separating hyperplane argument, there must exist some $n$ (and therefore some $z_n$) s.t. $P_n \in \mathtt{convhull}(\chi)$ and $P_{n+1} \notin \mathtt{convhull}(\chi)$. This contradicts the assumption that the likelihood model is SWC. Therefore any SWC model is also WAC. We demonstrate that the converse is not true by counterexample in Section 4.2.2.

### 4.2.1 WAC for static models

Analogous to (3), we define

$$T_j(\gamma) = \sum_i \gamma_i \log \frac{\alpha_{ij}}{\beta_{ij}}. \tag{6}$$

For a given $\gamma$, define the logical vector $r(\gamma)$ as

$$r_j(\gamma) = \left\{ \begin{array}{ll} 0 & T_j(\gamma) < 0 \\ 1 & T_j(\gamma) > 0 \\ \text{undet} & T_j(\gamma) = 0 \end{array} \right. \tag{7}$$

**Lemma 2** *A likelihood model is WAC if $\forall \gamma$ s.t. $\lim_{n \to \infty} P_n \in \{0,1\}^N$, $\exists i$ s.t. $r(\gamma) = \chi e_i$.*

Define the sets $\mathcal{P}_i = \{\gamma | r(\gamma) = \chi e_i\}$. Lemma 2 states that for a WAC likelihood model, $\{\mathcal{P}_i\}$ partitions the simplex (excluding unstable edge events) into sets of distributions s.t. $\gamma \in \mathcal{P}_i \Rightarrow P_n \to \chi e_i$. It is simple to show that the sets $\mathcal{P}_i$ are convex, and by definition the boundaries between sets are linear.

### 4.2.2 Motivating Example Revisited

Consider again the motivating example of the two IDPSs from Section 1.1. Recall that $\texttt{IDPS}_1$ detects a superset of the attacks detected by $\texttt{IDPS}_2$, and so this scenario conforms to the characteristic matrix analyzed in Section 4.1. Therefore (5) gives necessary and sufficient conditions for SWC, while (7) gives necessary and sufficient conditions for WAC.

Suppose that both the IDPSs use the interval between packet arrivals as their observation and assume the learned likelihood models for the two IDPSs happen to be geometrically distributed with parameters $x_1, x_2$ (when an attack is occurring) and $y_1, y_2$ (when no attack is occurring), with the index denoting the IDPS. We will analyze SWC and WAC for this class of models.

Plugging the given likelihood model into (5) implies that the model is SWC iff, for $z = 0, 1, 2, \ldots$

$$\left(\frac{1 - x_1}{1 - y_1}\right)^z \frac{x_1}{y_1} \geq \left(\frac{1 - x_2}{1 - y_2}\right)^z \frac{x_2}{y_2} \tag{8}$$

Equation (8) will be satisfied iff $\frac{x_1}{y_1} \geq \frac{x_2}{y_2}$ and $\frac{1 - x_1}{1 - y_1} \geq \frac{1 - x_2}{1 - y_2}$, which is therefore a necessary and sufficient condition for SWC.

Now, we turn to WAC. Forming $T$ as defined in (6), we see that

$$T_j(\gamma) = \sum_z \gamma_z z \log \frac{1 - x_j}{1 - y_j} + \log \frac{x_j}{y_j} = \mu \log \frac{1 - x_j}{1 - y_j} + \log \frac{x_j}{y_j} \tag{9}$$

where $\mu = E_\gamma[z]$. By the structure of the characteristic matrix, the model will be WAC iff $T_2(\gamma) > 0 \Rightarrow T_1(\gamma) > 0$ for all $\mu \geq 0$. Assume for convenience that $x_i > y_i$. Then $\{\gamma | T_i(\gamma) < 0\} = \{\gamma | \mu < \frac{\log y_i / x_i}{\log (1 - x_i)/(1 - y_i)}\}$ and therefore the model is WAC iff

$$\frac{\log \frac{x_2}{y_2}}{\log \frac{1 - x_2}{1 - y_2}} \geq \frac{\log \frac{x_1}{y_1}}{\log \frac{1 - x_1}{1 - y_1}} \tag{10}$$

Comparing the conditions for SWC (8) to those for WAC (10), we see that any parameters satisfying (8) also satisfy (10) but not vice versa. For example $x_1 = 0.3$, $x_2 = 0.5$, $y_1 = 0.2$, $y_2 = 0.25$ don't satisfy (8), but do satisfy (10). Thus WAC is truly a weaker sense of convergence than SWC.

## 5 Coherence with only finitely many observations

As shown in Sections 3 and 4, a WAC likelihood model generates a partition $\{\mathcal{P}_i\}$ over the observation probability simplex such that $\gamma \in \mathcal{P}_i \Rightarrow P_n \to \chi e_i$. The question we now address is, given a WAC likelihood model and finitely many observations (with empirical distribution $\hat{\gamma}_n$), how to revise an incoherent posterior probability assessment $P_n$ so that it is both coherent and consistent with the observed data.

> **Principle of Conserving Predictive Uncertainty**: Given $\hat{\gamma}_n$, choose $\lambda$ such that $\lambda_i = \Pr[\lim_{n \to \infty} \hat{\gamma}_n \in \mathcal{P}_i]$ for each $i$ (where $\gamma \in \mathcal{P}_i$ iff $P_n \to \chi e_i$).

The principle of conserving predictive uncertainty states that in revising an incoherent assessment $P_n$ to a coherent one $\tilde{P}_n$, the weight vectors over the columns of $\chi$ should reflect the uncertainty in whether the observations are being generated by a distribution in the corresponding element of the partition $\{\mathcal{P}_i\}$ (and therefore whether $P_n$ is converging to $\chi e_i$).

Given a uniform prior over generating distributions $\gamma$ and assuming Lebesgue measure $\mu$ over the parameters of the generating distribution, we can write

$$
\begin{aligned}
P(\gamma \in \mathcal{P}_i | \hat{\gamma}_n) &= \int_{\gamma \in \mathcal{P}_i} P(\gamma | \hat{\gamma}_n) d\mu &&= \int_{\gamma \in \mathcal{P}_i} \frac{P(\hat{\gamma}_n | \gamma) P(\gamma)}{\int_{\mathcal{P}} P(\hat{\gamma}_n | \gamma') P(\gamma') d\mu'} d\mu \\
&= \int_{\gamma \in \mathcal{P}_i} \frac{P(\hat{\gamma}_n | \gamma)}{\int_{\mathcal{P}} P(\hat{\gamma}_n | \gamma') d\mu'} d\mu &&= \frac{1}{\int_{\mathcal{P}} P(\hat{\gamma}_n | \gamma') d\mu'} \int_{\gamma \in \mathcal{P}_i} P(\hat{\gamma}_n | \gamma) d\mu
\end{aligned}
$$

In the limit of large $n$ $P(\hat{\gamma}_n | \gamma) \doteq e^{-nD(\hat{\gamma} || \gamma)}$ (where $\doteq$ denotes equality to the first degree in the exponent; c.f. [15]). This implies that as $n$ gets large, $\Pr[\lim_{n \to \infty} \hat{\gamma}_n \in \mathcal{P}_i]$ is dominated by the point $\gamma_i^* = \operatorname{argmin}_{\gamma \in \mathcal{P}_i} D(\hat{\gamma}_n || \gamma)$ (i.e. the reverse i-projection, or Maximum Likelihood estimate). This suggests the following approximation method for determining a coherent projection of $P_n$:

$$
\lambda_j = \frac{P(\hat{\gamma} | \gamma_i^*)}{\sum_{j \le |\{\mathcal{P}_i\}|} P(\hat{\gamma} | \gamma_j^*)} \tag{11}
$$

The relationship between the ML estimates ($\gamma_i^*$) and the probability over the columns of the characteristic matrix is represented graphically in Figure 1. As will be shown in Section 6, the principle of conserving predictive uncertainty can even be effectively applied to non-WAC models.

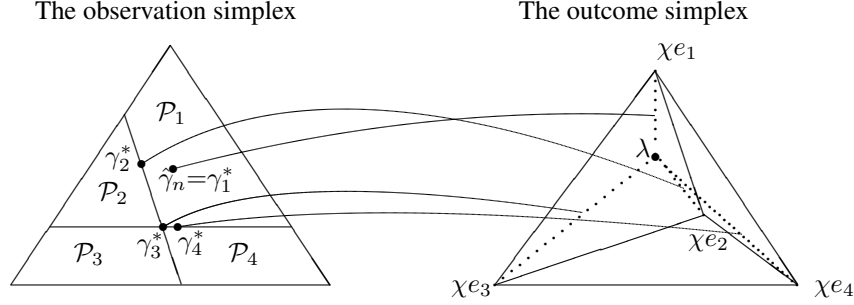

Figure 1: The relationship between observation and outcome simplices

## 5.1 Sparse coherent approximation

In general $|\Theta|$ (the length of the vector $\lambda$) can be of order $2^N$ (where $N$ is the number of assessors), so solving for $\lambda$ directly using (11) may be computationally infeasible. The following result suggests that to generate the optimal (in the sense of capturing to most possible weight) $O(N)$ sparse approximation of $\lambda$ we need only calculate the $O(N^2)$ reverse i-projections.

Let $\lambda$ be determined according to (11) and let $\{\mathcal{P}_i\}$ be as defined in Section 4. Assume wlog that $\lambda_i \ge \lambda_j$ for all $i > j$. Define the neighborhood of $\mathcal{P}_i$ as $\mathcal{N}(\mathcal{P}_i) = \{\mathcal{P}_j : |r(\mathcal{P}_i) - r(\mathcal{P}_j)| = 1\}$ where $r(\mathcal{P}_i)$ is defined as in (7). The neighborhood of $\mathcal{P}_i$ is the set of partition elements such that the limit of one (and only one) assessor's probability assessment has changed. The size of the neighborhood is thus less than or equal to $N$.

By the assumed ordering of $\lambda$ and (11), it is immediately evident that $\hat{\gamma} = \gamma_1^*$, i.e. the maximally weighted partition element is the one that contains the empirical distribution. It can be shown that $\gamma_2^* \in \mathcal{N}(\mathcal{P}_1)$, and thus recursively that $\gamma_i^* \in \bigcup_{j < i} \mathcal{N}(\mathcal{P}_j)$. Therefore the total number of projections in calculating the $i = N$ largest weights is bounded by

$$
\left| \bigcup_{j < i} \mathcal{N}(\mathcal{P}_j) \right| \le \sum_{j < i} |\mathcal{N}(\mathcal{P}_j)| \le \sum_{j < i} \max_j |\mathcal{N}(\mathcal{P}_j)| \le \sum_{j < i} N = N^2.
$$

# 6 Experimental Results

Consider a three-assessor situation with an identity characteristic matrix, i.e. each of three assessors estimates the probability that his unique outcome has occured knowing exactly one has occurred. Suppose each event is *a priori* equally likely, and a sequence of iid observations is generated with conditional probability $p(z^i|A^i) = 0.4$ and $p(z^{\bar{i}}|A^i) = 0.3$ (thus observation $z^i$ is evidence that event $A^i$ has occurred). Optimal joint estimation results in the posterior distribution convergence regions shown in Figure 6(a). Marginal estimation introduces incoherent convergence regions (6(b)); but for well-calibrated models, the empirical distribution is exponentially unlikely to lie in an incoherent region. However, miscalibrated models (6(c)) may lead to the true distribution lying in an incoherence region. WAC-approximation can ameliorate such miscalibration. The results of a

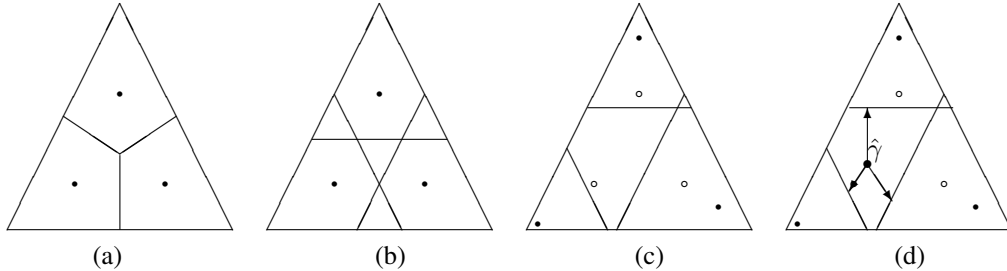

| (a) | (b) | (c) | (d) |

Figure 2: (a) Decision boundaries for optimal joint estimator; (b) Decision boundaries for marginal estimator; (c) Decision boundaries for miscalibrated observation model; (d) WAC approximation

Monte Carlo implementation of this miscalibrated estimation is shown in Figure 3. The top line (blue) shows the average error for accepting the posterior assessments generated by the miscalibrated observation models. The next line (green) corresponds to renormalization at each time step, equivalent to projecting the posterior into the coherent set with a divergence-based objective function. Next (red) shows the error generated by standard (L2) projection of the miscalibrated posterior into the coherent set. Finally, in cyan is shown the WAC approximation.

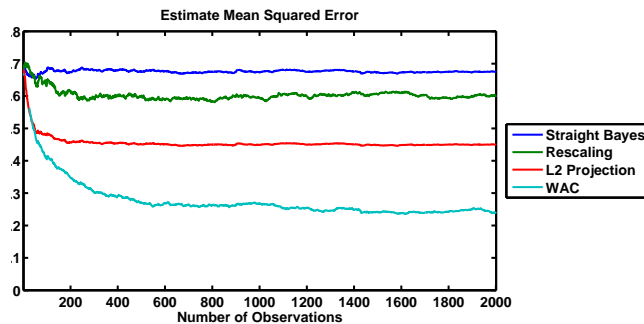

Figure 3: Comparison of mean-square errors as a function of the number of observations under four different estimation techniques

# 7 Conclusions

This paper has introduced the problem of *dynamic coherence* and analyzed it when the dynamics are induced by Bayes' rule. First, we demonstrated how under subjective event likelihood models (potentially unmatched to the true underlying distributions) Bayes' rule results in a partition over the observation probability simplex. Then we introduced two concepts of dynamic coherence: step-wise coherence and weak, asymptotic coherence. Next we suggested a principle of conservation of predictive uncertainty, by which observation-based incoherence can be mitigated (even in incoherent models). Finally, we briefly analyzed the computational impact of coherent approximation.

# References

[1] V.S. Borkar, V.R. Konda, and S.K. Mitter. On De Finetti coherence and Kolmogorov probability. *Statistics and Probability Letters*, 66(4):417–421, March 2004.

[2] Bruno de Finetti. *Theory of Probability*, volume 1-2. Wiley New York, 1974.

[3] Daniel Kahneman, Paul Slovic, and Amos Tversky, editors. *Judgment under uncertainty: Heuristics and biases*. Cambridge University Press, 1982.

[4] J.B. Predd *et al*. Aggregating forecasts of chance from incoherent and abstaining experts. *Decision Analysis*, 5:177–189, 2008.

[5] D.N. Osherson and M.Y. Vardi. Aggregating disparate estimates of chance. *Games and Economic Behavior*, 56(1):148–173, July 2006.

[6] P. Jones, S. Mitter, and V. Saligrama. Revision of marginal probability assessments. In *the $13^{th}$ Internationl Conference on Information Fusion*, Edinburgh, UK, 2010.

[7] K. Scarfone and P. Mell. Guide to intrusion detection and prevention systems (IDPS). Technical Report 800-94, National Institute of Standards and Technology, Technology Administration, US Dept. of Commerce.

[8] D.A. Freedman and R.A. Purves. Bayes' method for bookies. *The Annals of Mathematical Statistics*, 40(4):1177–1186, August 1969.

[9] D. Heath and W. Sudderth. On finitely additive priors, coherence, and extended admissibility. *The Annals of Statistics*, 6(2):333–345, March 1978.

[10] D.A. Lane and W. Sudderth. Coherent and continuous inference. *The Annals of Statistics*, 11(1):114–120, March 1983.

[11] E. Regazzini. De Finetti's coherence and statistical inference. *The Annals of Statistics*, 15(2):845–864, June 1987.

[12] E. Regazzini. Coherent statistical inference and bayes theorem. *The Annals of Statistics*, 19(1):366–381, March 1991.

[13] P. Diaconis and S.L. Zabell. Updating subjective probability. *Journal of the American Statistical Association*, 77(380):822–830, December 1982.

[14] H. Chan and A. Darwiche. On the revision of probabilistic beliefs using uncertain evidence. *Artificial Intelligence*, 40(4):67–90, August 2005.

[15] Joy Thomas and Thomas Cover. *Elements of Information Theory*. Wiley Interscience, 2nd edition, 2006.

[16] M. Feder and N. Merhav. Universal composite hypothesis testing: A competitive minimax approach. *IEEE Transactions on Information Theory*, 48(6):1504–1517, June 2002.

